# Multi-Instance Multi-Label Learning with Application to Scene Classification

**Zhi-Hua Zhou     Min-Ling Zhang**
National Laboratory for Novel Software Technology
Nanjing University, Nanjing 210093, China
{zhouzh,zhangml}@lamda.nju.edu.cn

## Abstract

In this paper, we formalize *multi-instance multi-label learning*, where each training example is associated with not only multiple instances but also multiple class labels. Such a problem can occur in many real-world tasks, e.g. an image usually contains multiple patches each of which can be described by a feature vector, and the image can belong to multiple categories since its semantics can be recognized in different ways. We analyze the relationship between multi-instance multi-label learning and the learning frameworks of *traditional supervised learning*, *multi-instance learning* and *multi-label learning*. Then, we propose the MIMLBOOST and MIMLSVM algorithms which achieve good performance in an application to scene classification.

## 1   Introduction

In *traditional supervised learning*, an object is represented by an instance (or feature vector) and associated with a class label. Formally, let $\mathcal{X}$ denote the instance space (or feature space) and $\mathcal{Y}$ the set of class labels. Then the task is to learn a function $f : \mathcal{X} \rightarrow \mathcal{Y}$ from a given data set $\{(\boldsymbol{x}_1, y_1), (\boldsymbol{x}_2, y_2), \cdots, (\boldsymbol{x}_m, y_m)\}$, where $\boldsymbol{x}_i \in \mathcal{X}$ is an instance and $y_i \in \mathcal{Y}$ the known label of $\boldsymbol{x}_i$.

Although the above formalization is prevailing and successful, there are many real-world problems which do not fit this framework well, where a real-world object may be associated with a number of instances and a number of labels simultaneously. For example, an image usually contains multiple patches each can be represented by an instance, while in image classification such an image can belong to several classes simultaneously, e.g. an image can belong to *mountains* as well as *Africa*. Another example is text categorization, where a document usually contains multiple sections each of which can be represented as an instance, and the document can be regarded as belonging to different categories if it was viewed from different aspects, e.g. a document can be categorized as *scientific novel*, *Jules Verne's writing* or even *books on travelling*. Web mining is a further example, where each of the links can be regarded as an instance while the web page itself can be recognized as *news page*, *sports page*, *soccer page*, etc.

In order to deal with such problems, in this paper we formalize *multi-instance multi-label learning* (abbreviated as MIML). In this learning framework, a training example is described by multiple instances and associated with multiple class labels. Formally, let $\mathcal{X}$ denote the instance space and $\mathcal{Y}$ the set of class labels. Then the task is to learn a function $f_{MIML} : 2^{\mathcal{X}} \rightarrow 2^{\mathcal{Y}}$ from a given data set $\{(X_1, Y_1), (X_2, Y_2), \cdots, (X_m, Y_m)\}$, where $X_i \subseteq \mathcal{X}$ is a set of instances $\{\boldsymbol{x}_1^{(i)}, \boldsymbol{x}_2^{(i)}, \cdots, \boldsymbol{x}_{n_i}^{(i)}\}$, $\boldsymbol{x}_j^{(i)} \in \mathcal{X}$ ($j = 1, 2, \cdots, n_i$), and $Y_i \subseteq \mathcal{Y}$ is a set of labels $\{y_1^{(i)}, y_2^{(i)}, \cdots, y_{l_i}^{(i)}\}$, $y_k^{(i)} \in \mathcal{Y}$ ($k = 1, 2, \cdots, l_i$). Here $n_i$ denotes the number of instances in $X_i$ and $l_i$ the number of labels in $Y_i$.

After analyzing the relationship between MIML and the frameworks of traditional supervised learning, *multi-instance learning* and *multi-label learning*, we propose two MIML algorithms, MIML-

BOOST and MIMLSVM. Application to scene classification shows that, solving some real-world problems in the MIML framework can achieve better performance than solving them in existing frameworks such as multi-instance learning and multi-label learning.

## 2 Multi-Instance Multi-Label Learning

We start by investigating the relationship between MIML and the frameworks of traditional supervised learning, multi-instance learning and multi-label learning, and then we develop some solutions.

Multi-instance learning [4] studies the problem where a real-world object described by a number of instances is associated with one class label. Formally, the task is to learn a function $f_{MIL} : 2^{\mathcal{X}} \to \{-1, +1\}$ from a given data set $\{(X_1, y_1), (X_2, y_2), \cdots, (X_m, y_m)\}$, where $X_i \subseteq \mathcal{X}$ is a set of instances $\{\boldsymbol{x}_1^{(i)}, \boldsymbol{x}_2^{(i)}, \cdots, \boldsymbol{x}_{n_i}^{(i)}\}$, $\boldsymbol{x}_j^{(i)} \in \mathcal{X}$ $(j = 1, 2, \cdots, n_i)$, $y_i \in \{-1, +1\}$ is the label of $X_i$.[1] Multi-instance learning techniques have been successfully applied to diverse applications including scene classification [3, 7].

Multi-label learning [8] studies the problem where a real-world object described by one instance is associated with a number of class labels. Formally, the task is to learn a function $f_{MLL} : \mathcal{X} \to 2^{\mathcal{Y}}$ from a given data set $\{(\boldsymbol{x}_1, Y_1), (\boldsymbol{x}_2, Y_2), \cdots, (\boldsymbol{x}_m, Y_m)\}$, where $\boldsymbol{x}_i \in \mathcal{X}$ is an instance and $Y_i \subseteq \mathcal{Y}$ a set of labels $\{y_1^{(i)}, y_2^{(i)}, \cdots, y_{l_i}^{(i)}\}$, $y_k^{(i)} \in \mathcal{Y}$ $(k = 1, 2, \cdots, l_i)$.[2] Multi-label learning techniques have also been successfully applied to scene classification [1].

In fact, the *multi-* learning frameworks result from the ambiguity in representing real-world objects. Multi-instance learning studies the ambiguity in the input space (or instance space), where an object has many alternative input descriptions, i.e. instances; multi-label learning studies the ambiguity in the output space (or label space), where an object has many alternative output descriptions, i.e. labels; while MIML considers the ambiguity in the input and output spaces simultaneously. We illustrate the differences among these learning frameworks in Figure 1.

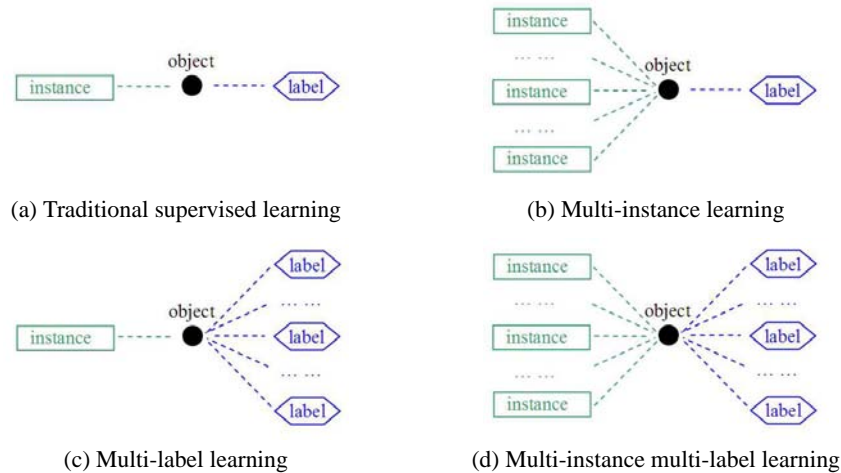

(a) Traditional supervised learning      (b) Multi-instance learning

(c) Multi-label learning      (d) Multi-instance multi-label learning

Figure 1: Four different learning frameworks

Traditional supervised learning is evidently a degenerated version of multi-instance learning as well as a degenerated version of multi-label learning, while traditional supervised learning, multi-instance learning and multi-label learning are all degenerated versions of MIML. Thus, we can tackle MIML by identifying its equivalence in the traditional supervised learning framework, using multi-instance learning or multi-label learning as the bridge.

**Solution 1**: Using multi-instance learning as the bridge: We can transform a MIML learning task, i.e. to learn a function $f_{MIML} : 2^{\mathcal{X}} \to 2^{\mathcal{Y}}$, into a multi-instance learning task, i.e. to learn a function $f_{MIL} : 2^{\mathcal{X}} \times \mathcal{Y} \to \{-1, +1\}$. For any $y \in \mathcal{Y}$, $f_{MIL}(X_i, y) = +1$ if $y \in Y_i$ and $-1$ otherwise. The proper labels for a new example $X^*$ can be determined according to $Y^* = \{y | \arg_{y \in \mathcal{Y}}[f_{MIL}(X^*, y) = +1]\}$. We can transform this multi-instance learning task further into a traditional supervised learning task, i.e. to learn a function $f_{SISL} : \mathcal{X} \times \mathcal{Y} \to \{-1, +1\}$, under a constraint specifying how to derive $f_{MIL}(X_i, y)$ from $f_{SISL}(\boldsymbol{x}_j^{(i)}, y)$ ($j = 1, \cdots, n_i$). For any $y \in \mathcal{Y}$, $f_{SISL}(\boldsymbol{x}_j^{(i)}, y) = +1$ if $y \in Y_i$ and $-1$ otherwise. Here the constraint can be $f_{MIL}(X_i, y) = sign[\sum_{j=1}^{n_i} f_{SISL}(\boldsymbol{x}_j^{(i)}, y)]$ which has been used in transforming multi-instance learning tasks into traditional supervised learning tasks [9].[3] Note that other kinds of constraint can also be used here.

**Solution 2**: Using multi-label learning as the bridge: We can also transform a MIML learning task, i.e. to learn a function $f_{MIML} : 2^{\mathcal{X}} \to 2^{\mathcal{Y}}$, into a multi-label learning task, i.e. to learn a function $f_{MLL} : \mathcal{Z} \to 2^{\mathcal{Y}}$. For any $\boldsymbol{z}_i \in \mathcal{Z}$, $f_{MLL}(\boldsymbol{z}_i) = f_{MIML}(X_i)$ if $\boldsymbol{z}_i = \phi(X_i)$, $\phi : 2^{\mathcal{X}} \to \mathcal{Z}$. The proper labels for a new example $X^*$ can be determined according to $Y^* = f_{MLL}(\phi(X^*))$. We can transform this multi-label learning task further into a traditional supervised learning task, i.e. to learn a function $f_{SISL} : \mathcal{Z} \times \mathcal{Y} \to \{-1, +1\}$. For any $y \in \mathcal{Y}$, $f_{SISL}(\boldsymbol{z}_i, y) = +1$ if $y \in Y_i$ and $-1$ otherwise. That is, $f_{MLL}(\boldsymbol{z}_i) = \{y | \arg_{y \in \mathcal{Y}}[f_{SISL}(\boldsymbol{z}_i, y) = +1]\}$. Here the mapping $\phi$ can be implemented with *constructive clustering* which has been used in transforming multi-instance bags into traditional single-instances [11]. Note that other kinds of mapping can also be used here.

## 3 Algorithms

In this section, we propose two algorithms for solving MIML problems: MIMLBOOST works along the first solution described in Section 2, while MIMLSVM works along the second solution.

### 3.1 MIMLBOOST

Given any set $\Omega$, let $|\Omega|$ denote its size, i.e. the number of elements in $\Omega$; given any predicate $\pi$, let $[\![\pi]\!]$ be 1 if $\pi$ holds and 0 otherwise; given $(X_i, Y_i)$, for any $y \in \mathcal{Y}$, let $\Psi(X_i, y) = +1$ if $y \in Y_i$ and $-1$ otherwise, where $\Psi$ is a function $\Psi : 2^{\mathcal{X}} \times \mathcal{Y} \to \{-1, +1\}$. The MIMLBOOST algorithm is presented in Table 1.

In the first step, each MIML example $(X_u, Y_u)$ ($u = 1, 2, \cdots, m$) is transformed into a set of $|\mathcal{Y}|$ number of multi-instance bags, i.e. $\{[(X_u, y_1), \Psi(X_u, y_1)], [(X_u, y_2), \Psi(X_u, y_2)], \cdots, [(X_u, y_{|\mathcal{Y}|}), \Psi(X_u, y_{|\mathcal{Y}|})]\}$. Note that $[(X_u, y_v), \Psi(X_u, y_v)]$ ($v = 1, 2, \cdots, |\mathcal{Y}|$) is a labeled multi-instance bag where $(X_u, y_v)$ is a bag containing $n_u$ number of instances, i.e. $\{(\boldsymbol{x}_1^{(u)}, y_v), (\boldsymbol{x}_2^{(u)}, y_v), \cdots, (\boldsymbol{x}_{n_u}^{(u)}, y_v)\}$, and $\Psi(X_u, y_v) \in \{+1, -1\}$ is the label of this bag.

Thus, the original MIML data set is transformed into a multi-instance data set containing $m \times |\mathcal{Y}|$ number of bags, i.e. $\{[(X_1, y_1), \Psi(X_1, y_1)], \cdots, [(X_1, y_{|\mathcal{Y}|}), \Psi(X_1, y_{|\mathcal{Y}|})], [(X_2, y_1), \Psi(X_2, y_1)], \cdots, [(X_m, y_{|\mathcal{Y}|}), \Psi(X_m, y_{|\mathcal{Y}|})]\}$. Let $[(X^{(i)}, y^{(i)}), \Psi(X^{(i)}, y^{(i)})]$ denote the $i$th of these $m \times |\mathcal{Y}|$ number of bags, that is, $(X^{(1)}, y^{(1)})$ denotes $(X_1, y_1), \cdots, (X^{(|\mathcal{Y}|)}, y^{(|\mathcal{Y}|)})$ denotes $(X_1, y_{|\mathcal{Y}|}), \cdots, (X^{(m \times |\mathcal{Y}|)}, y^{(m \times |\mathcal{Y}|)})$ denotes $(X_m, y_{|\mathcal{Y}|})$, where $(X^{(i)}, y^{(i)})$ contains $n_i$ number of instances, i.e. $\{(\boldsymbol{x}_1^{(i)}, y^{(i)}), (\boldsymbol{x}_2^{(i)}, y^{(i)}), \cdots, (\boldsymbol{x}_{n_i}^{(i)}, y^{(i)})\}$.

Then, from the data set a multi-instance learning function $f_{MIL}$ can be learned, which can accomplish the desired MIML function because $f_{MIML}(X^*) = \{y | \arg_{y \in \mathcal{Y}}(sign[f_{MIL}(X^*, y)] = +1)\}$. Here we use MIBOOSTING [9] to implement $f_{MIL}$.

For convenience, let $(B, g)$ denote the bag $[(X, y), \Psi(X, y)]$. Then, here the goal is to learn a function $\mathcal{F}(B)$ minimizing the bag-level exponential loss $E_{\mathcal{B}} E_{\mathcal{G}|\mathcal{B}}[\exp(-g\mathcal{F}(B))]$, which ultimately

Table 1: The MIMLBOOST algorithm

---

1  Transform each MIML example $(X_u, Y_u)$ $(u = 1, 2, \cdots, m)$ into $|\mathcal{Y}|$ number of multi-instance bags $\{[(X_u, y_1), \Psi(X_u, y_1)], \cdots, [(X_u, y_{|\mathcal{Y}|}), \Psi(X_u, y_{|\mathcal{Y}|})]\}$. Thus, the original data set is transformed into a multi-instance data set containing $m \times |\mathcal{Y}|$ number of multi-instance bags, denoted by $\{[(X^{(i)}, y^{(i)}), \Psi(X^{(i)}, y^{(i)})]\}$ $(i = 1, 2, \cdots, m \times |\mathcal{Y}|)$.

2  Initialize weight of each bag to $W^{(i)} = \frac{1}{m \times |\mathcal{Y}|}$ $(i = 1, 2, \cdots, m \times |\mathcal{Y}|)$.

3  Repeat for $t = 1, 2, \cdots, T$ iterations:

   3a  Set $W_j^{(i)} = W^{(i)}/n_i$ $(i = 1, 2, \cdots, m \times |\mathcal{Y}|)$, assign the bag's label $\Psi(X^{(i)}, y^{(i)})$ to each of its instances $(\boldsymbol{x}_j^{(i)}, y^{(i)})$ $(j = 1, 2, \cdots, n_i)$, and build an instance-level predictor $h_t[(\boldsymbol{x}_j^{(i)}, y^{(i)})] \in \{-1, +1\}$.

   3b  For the $i$th bag, compute the error rate $e^{(i)} \in [0, 1]$ by counting the number of misclassified instances within the bag, i.e. $e^{(i)} = \frac{\sum_{j=1}^{n_i} [\![h_t[(\boldsymbol{x}_j^{(i)}, y^{(i)})] \neq \Psi(X^{(i)}, y^{(i)})]\!]}{n_i}$.

   3c  If $e^{(i)} < 0.5$ for all $i \in \{1, 2, \cdots, m \times |\mathcal{Y}|\}$, go to Step 4.

   3d  Compute $c_t = \arg\min_{c_t} \sum_{i=1}^{m \times |\mathcal{Y}|} W^{(i)} \exp[(2e^{(i)} - 1)c_t]$.

   3e  If $c_t \leq 0$, go to Step 4.

   3f  Set $W^{(i)} = W^{(i)} \exp[(2e^{(i)} - 1)c_t]$ $(i = 1, 2, \cdots, m \times |\mathcal{Y}|)$ and re-normalize such that $0 \leq W^{(i)} \leq 1$ and $\sum_{i=1}^{m \times |\mathcal{Y}|} W^{(i)} = 1$.

4  Return $Y^* = \{y | \arg_{y \in \mathcal{Y}} sign\left(\sum_j \sum_t c_t h_t[(\boldsymbol{x}_j^*, y)]\right) = +1\}$ ($\boldsymbol{x}_j^*$ is $X^*$'s $j$th instance).

---

estimates the bag-level log-odds function $\frac{1}{2} \log \frac{Pr(g=1|B)}{Pr(g=-1|B)}$. In each boosting round, the aim is to expand $\mathcal{F}(B)$ into $\mathcal{F}(B) + cf(B)$, i.e. adding a new weak classifier, so that the exponential loss is minimized. Assuming all instances in a bag contribute equally and independently to the bag's label, $f(B) = \frac{1}{n_B} \sum_j h(\boldsymbol{b}_j)$ can be derived, where $h(\boldsymbol{b}_j) \in \{-1, +1\}$ is the prediction of the instance-level classifier $h(\cdot)$ for the $j$th instance in bag $B$, and $n_B$ is the number of instances in $B$.

It has been shown by [9] that the best $f(B)$ to be added can be achieved by seeking $h(\cdot)$ which maximizes $\sum_i \sum_{j=1}^{n_i} [\frac{1}{n_i} W^{(i)} g^{(i)} h(\boldsymbol{b}_j^{(i)})]$, given the bag-level weights $W = \exp(-g\mathcal{F}(B))$. By assigning each instance the label of its bag and the corresponding weight $W^{(i)}/n_i$, $h(\cdot)$ can be learned by minimizing the weighted instance-level classification error. This actually corresponds to the Step 3a of MIMLBOOST. When $f(B)$ is found, the best multiplier $c > 0$ can be got by directly optimizing the exponential loss:

$$E_{\mathcal{B}} E_{\mathcal{G}|\mathcal{B}}[\exp(-g\mathcal{F}(B) + c(-gf(B)))] = \sum_i W^{(i)} \exp[c\left(-\frac{g^{(i)} \sum_j h(\boldsymbol{b}_j^{(i)})}{n_i}\right)]$$

$$= \sum_i W^{(i)} \exp[(2e^{(i)} - 1)c]$$

where $e^{(i)} = \frac{1}{n_i} \sum_j [\![(h(\boldsymbol{b}_j^{(i)}) \neq g^{(i)})]\!]$ (computed in Step 3b). Minimization of this expectation actually corresponds to Step 3d, where numeric optimization techniques such as quasi-Newton method can be used. Finally, the bag-level weights are updated in Step 3f according to the additive structure of $\mathcal{F}(B)$.

## 3.2 MIMLSVM

Given $(X_i, Y_i)$ and $\boldsymbol{z}_i = \phi(X_i)$ where $\phi : 2^{\mathcal{X}} \rightarrow \mathcal{Z}$, for any $y \in \mathcal{Y}$, let $\Phi(\boldsymbol{z}_i, y) = +1$ if $y \in Y_i$ and $-1$ otherwise, where $\Phi$ is a function $\Phi : \mathcal{Z} \times \mathcal{Y} \rightarrow \{-1, +1\}$. The MIMLSVM algorithm is presented in Table 2.

In the first step, the $X_u$ of each MIML example $(X_u, Y_u)$ $(u = 1, 2, \cdots, m)$ is collected and put into a data set $\Gamma$. Then, in the second step, $k$-medoids clustering is performed on $\Gamma$. Since each

Table 2: The MIMLSVM algorithm

---

1   For MIML examples $(X_u, Y_u)$ $(u = 1, 2, \cdots, m)$, $\Gamma = \{X_u | u = 1, 2, \cdots, m\}$.

2   Randomly select $k$ elements from $\Gamma$ to initialize the medoids $M_t$ $(t = 1, 2, \cdots, k)$,
    repeat until all $M_t$ do not change:
    2a   $\Gamma_t = \{M_t\}$ $(t = 1, 2, \cdots, k)$.
    2b   Repeat for each $X_u \in (\Gamma - \{M_t | t = 1, 2, \cdots, k\})$:
         $index = \arg\min_{t \in \{1, \cdots, k\}} d_H(X_u, M_t)$, $\Gamma_{index} = \Gamma_{index} \cup \{X_u\}$.
    2c   $M_t = \arg\min_{A \in \Gamma_t} \sum_{B \in \Gamma_t} d_H(A, B)$ $(t = 1, 2, \cdots, k)$.

3   Transform $(X_u, Y_u)$ into a multi-label example $(\boldsymbol{z}_u, Y_u)$ $(u = 1, 2, \cdots, m)$, where
    $\boldsymbol{z}_u = (\boldsymbol{z}_{u1}, \boldsymbol{z}_{u2}, \cdots, \boldsymbol{z}_{uk}) = (d_H(X_u, M_1), d_H(X_u, M_2), \cdots, d_H(X_u, M_k))$.

4   For each $y \in \mathcal{Y}$, derive a data set $\mathcal{D}_y = \{(\boldsymbol{z}_u, \Phi(\boldsymbol{z}_u, y)) | u = 1, 2, \cdots, m\}$, and then
    train an SVM $h_y = SVMTrain(\mathcal{D}_y)$.

5   Return $Y^* = \{\arg\max_{y \in \mathcal{Y}} h_y(\boldsymbol{z}^*)\} \cup \{y | h_y(\boldsymbol{z}^*) \geq 0, y \in \mathcal{Y}\}$, where $\boldsymbol{z}^* = (d_H(X^*, M_1),$
    $d_H(X^*, M_2), \cdots, d_H(X^*, M_k))$.

---

data item in $\Gamma$, i.e. $X_u$, is an unlabeled multi-instance bag instead of a single instance, we employ Hausdorff distance [5] to measure the distance. In detail, given two bags $A = \{\boldsymbol{a}_1, \boldsymbol{a}_2, \cdots, \boldsymbol{a}_{n_A}\}$ and $B = \{\boldsymbol{b}_1, \boldsymbol{b}_2, \cdots, \boldsymbol{b}_{n_B}\}$, the Hausdorff distance between $A$ and $B$ is defined as

$$d_H(A, B) = \max\{\max_{\boldsymbol{a} \in A} \min_{\boldsymbol{b} \in B} \|\boldsymbol{a} - \boldsymbol{b}\|, \max_{\boldsymbol{b} \in B} \min_{\boldsymbol{a} \in A} \|\boldsymbol{b} - \boldsymbol{a}\|\}$$

where $\|\boldsymbol{a} - \boldsymbol{b}\|$ measures the distance between the instances $\boldsymbol{a}$ and $\boldsymbol{b}$, which takes the form of Euclidean distance here.

After the clustering process, we divide the data set $\Gamma$ into $k$ partitions whose medoids are $M_t$ $(t = 1, 2, \cdots, k)$, respectively. With the help of these medoids, we transform the original multi-instance example $X_u$ into a $k$-dimensional numerical vector $\boldsymbol{z}_u$, where the $i$th $(i = 1, 2, \cdots, k)$ component of $\boldsymbol{z}_u$ is the distance between $X_u$ and $M_i$, that is, $d_H(X_u, M_i)$. In other words, $\boldsymbol{z}_{ui}$ encodes some structure information of the data, that is, the relationship between $X_u$ and the $i$th partition of $\Gamma$. This process reassembles the *constructive clustering* process used by [11] in transforming multi-instance examples into single-instance examples except that in [11] the clustering is executed at the instance level while here we execute it at the bag level. Thus, the original MIML examples $(X_u, Y_u)$ $(u = 1, 2, \cdots, m)$ have been transformed into multi-label examples $(\boldsymbol{z}_u, Y_u)$ $(u = 1, 2, \cdots, m)$, which corresponds to the Step 3 of MIMLSVM. Note that this transformation may lose information, nevertheless the performance of MIMLSVM is still good. This suggests that MIML is a powerful framework which has captured more original information than other learning frameworks.

Then, from the data set a multi-label learning function $f_{MLL}$ can be learned, which can accomplish the desired MIML function because $f_{MIML}(X^*) = f_{MLL}(\boldsymbol{z}^*)$. Here we use MLSVM [1] to implement $f_{MLL}$.

Concretely, MLSVM decomposes the multi-label learning problem into multiple independent binary classification problems (one per class), where each example associated with the label set $Y$ is regarded as a positive example when building SVM for any class $y \in Y$, while regarded as a negative example when building SVM for any class $y \notin Y$, as shown in the Step 4 of MIMLSVM. In making predictions, the *T-Criterion* [1] is used, which actually corresponds to the Step 5 of the MIMLSVM algorithm. That is, the test example is labeled by all the class labels with positive SVM *scores*, except that when all the SVM scores are negative, the test example is labeled by the class label which is with the *top* (least negative) score.

## 4   Application to Scene Classification

The data set consists of 2,000 natural scene images belonging to the classes *desert*, *mountains*, *sea*, *sunset*, and *trees*, as shown in Table 3. Some images were from the COREL image collection while some were collected from the Internet. Over 22% images belong to multiple classes simultaneously.

Table 3: The image data set (*d*: *desert*, *m*: *mountains*, *s*: *sea*, *su*: *sunset*, *t*: *trees*)

| label | # images | label | # images | label | # images | label | # images |
|---|---|---|---|---|---|---|---|
| *d* | 340 | *d + m* | 19 | *m + su* | 19 | *d + m + su* | 1 |
| *m* | 268 | *d + s* | 5 | *m + t* | 106 | *d + su + t* | 3 |
| *s* | 341 | *d + su* | 21 | *s + su* | 172 | *m + s + t* | 6 |
| *su* | 216 | *d + t* | 20 | *s + t* | 14 | *m + su + t* | 1 |
| *t* | 378 | *m + s* | 38 | *su + t* | 28 | *s + su + t* | 4 |

## 4.1 Comparison with Multi-Label Learning Algorithms

Since the scene classification task has been successfully tackled by multi-label learning algorithms [1], we compare the MIML algorithms with established multi-label learning algorithms AD-ABOOST.MH [8] and MLSVM [1]. The former is the core of a successful multi-label learning system BOOSTEXTER [8], while the latter has achieved excellent performance in scene classification [1].

For MIMLBOOST and MIMLSVM, each image is represented as a bag of nine instances generated by the SBN method [7]. Here each instance actually corresponds to an image patch, and better performance can be expected with better image patch generation method. For ADABOOST.MH and MLSVM, each image is represented as a feature vector obtained by concatenating the instances of MIMLBOOST or MIMLSVM. Gaussian kernel LIBSVM [2] is used to implement MLSVM, where the *cross-training* strategy is used to build the classifiers while the *T-Criterion* is used to label the images [1]. The MIMLSVM algorithm is also realized with a Gaussian kernel, while the parameter $k$ is set to be 20% of the number of training images.[4] Note that the instance-level predictor used in Step 3a of MIMLBOOST is also a Gaussian kernel LIBSVM (with default parameters).

Since ADABOOST.MH and MLSVM make multi-label predictions, here the performance of the compared algorithms are evaluated according to five multi-label evaluation metrics, as shown in Tables 4 to 7, where '↓' indicates 'the smaller the better' while '↑' indicates 'the bigger the better'. Details of these evaluation metrics can be found in [8]. Tenfold cross-validation is performed and 'mean ± std' is presented in the tables, where the best performance achieved by each algorithm is bolded. Note that since in each boosting round MIMLBOOST performs more operations than ADABOOST.MH does, for fair comparison, the boosting rounds used by ADABOOST.MH are set to ten times of that used by MIMLBOOST such that the time cost of them are comparable.

Table 4: The performance of MIMLBOOST with different boosting rounds

| boosting | evaluation metric | | | | |
|---|---|---|---|---|---|
| rounds | $hamm.\ loss^\downarrow$ | $one\text{-}error^\downarrow$ | $coverage^\downarrow$ | $rank.\ loss^\downarrow$ | $ave.\ prec.^\uparrow$ |
| 5 | .202±.011 | .373±.045 | 1.026±.093 | .208±.028 | .764±.027 |
| 10 | .197±.010 | .362±.040 | 1.013±.109 | .191±.027 | .770±.026 |
| 15 | .195±.009 | .361±.034 | 1.004±.101 | .186±.025 | .772±.023 |
| 20 | .193±.008 | .355±.037 | .996±.102 | .183±.025 | .775±.024 |
| 25 | **.189±.009** | **.351±.039** | **.989±.103** | **.181±.026** | **.777±.025** |

Table 5: The performance of ADABOOST.MH with different boosting rounds

| boosting | evaluation metric | | | | |
|---|---|---|---|---|---|
| rounds | $hamm.\ loss^\downarrow$ | $one\text{-}error^\downarrow$ | $coverage^\downarrow$ | $rank.\ loss^\downarrow$ | $ave.\ prec.^\uparrow$ |
| 50 | **.228±.013** | .473±.031 | 1.299±.099 | .263±.022 | .695±.022 |
| 100 | .234±.019 | .465±.042 | 1.292±.138 | .259±.030 | .698±.033 |
| 150 | .233±.020 | .465±.053 | 1.279±.140 | .255±.032 | .700±.033 |
| 200 | .232±.012 | .453±.031 | 1.269±.107 | .253±.022 | .706±.020 |
| 250 | .231±.018 | **.451±.046** | **1.258±.137** | **.250±.031** | **.708±.030** |

Table 6: The performance of MIMLSVM with different $\gamma$ used in Gaussian kernel

| Gaussian | evaluation metric | | | | |
|---|---|---|---|---|---|
| kernel | $hamm.\ loss^{\downarrow}$ | $one\text{-}error^{\downarrow}$ | $coverage^{\downarrow}$ | $rank.\ loss^{\downarrow}$ | $ave.\ prec.^{\uparrow}$ |
| $\gamma = .1$ | .181±.017 | .332±.036 | 1.024±.089 | **.187±.018** | .780±.021 |
| $\gamma = .2$ | **.180±.017** | **.327±.033** | **1.022±.085** | **.187±.018** | **.783±.020** |
| $\gamma = .3$ | .188±.016 | .344±.032 | 1.065±.094 | .196±.020 | .772±.020 |
| $\gamma = .4$ | .193±.014 | .358±.030 | 1.080±.099 | .202±.022 | .764±.021 |
| $\gamma = .5$ | .196±.014 | .370±.033 | 1.109±.101 | .209±.023 | .757±.023 |

Table 7: The performance of MLSVM with different $\gamma$ used in Gaussian kernel

| Gaussian | evaluation metric | | | | |
|---|---|---|---|---|---|
| kernel | $hamm.\ loss^{\downarrow}$ | $one\text{-}error^{\downarrow}$ | $coverage^{\downarrow}$ | $rank.\ loss^{\downarrow}$ | $ave.\ prec.^{\uparrow}$ |
| $\gamma = 1$ | .200±.014 | .379±.032 | 1.125±.115 | .214±.020 | .751±.022 |
| $\gamma = 2$ | .196±.013 | **.368±.032** | **1.115±.122** | **.211±.023** | **.756±.022** |
| $\gamma = 3$ | **.195±.015** | .370±.034 | 1.129±.113 | .214±.022 | .754±.023 |
| $\gamma = 4$ | .196±.016 | .372±.034 | 1.151±.122 | .220±.024 | .751±.023 |
| $\gamma = 5$ | .202±.015 | .388±.032 | 1.181±.128 | .229±.026 | .741±.023 |

Comparing Tables 4 to 7 we can find that both MIMLBOOST and MIMLSVM are apparently better than ADABOOST.MH and MLSVM. Impressively, pair-wise $t$-tests with .05 significance level reveal that the worst performance of MIMLBOOST (with 5 boosting rounds) is even significantly better than the best performance of ADABOOST.MH (with 250 boosting rounds) on all the evaluation metrics, and is significantly better than the best performance of MLSVM (with $\gamma = 2$) in terms of *coverage* while comparable on the remaining metrics; the worse performance of MIMLSVM (with $\gamma = .5$) is even comparable to the best performance of MLSVM and is significantly better than the best performance of ADABOOST.MH on all the evaluation metrics. These observations confirm that formalizing the scene classification task as a MIML problem to solve by MIMLBOOST or MIMLSVM is better than formalizing it as a multi-label learning problem to solve by ADABOOST.MH or MLSVM.

### 4.2 Comparison with Multi-Instance Learning Algorithms

Since the scene classification task has been successfully tackled by multi-instance learning algorithms [7], we compare the MIML algorithms with established multi-instance learning algorithms DIVERSE DENSITY [7] and EM-DD [10]. The former is one of the most influential multi-instance learning algorithm and has achieved excellent performance in scene classification [7], while the latter has achieved excellent performance on multi-instance benchmark tests [10].

Here all the compared algorithms use the same input representation. That is, each image is represented as a bag of nine instances generated by the SBN method [7]. The parameters of DIVERSE DENSITY and EM-DD are set according to the settings that resulted in the best performance [7, 10]. The MIMLBOOST and MIMLSVM algorithms are implemented as described in Section 4.1, with 25 boosting rounds for MIMLBOOST while $\gamma = .2$ for MIMLSVM.

Since DIVERSE DENSITY and EM-DD make single-label predictions, here the performance of the compared algorithms are evaluated according to *predictive accuracy*, i.e. classification accuracy on test set. Note that for MIMLBOOST and MIMLSVM, the *top ranked class* is regarded as the single-label prediction. Tenfold cross-validation is performed and 'mean ± std' is presented in Table 8, where the best performance on each image class is bolded. Note that besides the predictive accuracies on each class, the overall accuracy is also presented, which is denoted by 'overall'.

We can find from Table 8 that MIMLBOOST achieves the best performance on image classes *desert* and *trees* while MIMLSVM achieves the best performance on the remaining image classes. Overall, MIMLSVM achieves the best performance. Pair-wise $t$-tests with .05 significance level reveal that the overall performance of MIMLSVM is comparable to that of MIMLBOOST, both are significantly better than that of DIVERSE DENSITY and EM-DD. These observations confirm that formalizing the scene classification task as a MIML problem to solve by MIMLBOOST or MIMLSVM is better than formalizing it as a multi-instance learning problem to solve by DIVERSE DENSITY or EM-DD.

Table 8: Compare predictive accuracy of MIMLBOOST, MIMLSVM, DIVERSE DENSITY and EM-DD

| Image | Compared algorithms | | | |
|---|---|---|---|---|
| class | MIMLBOOST | MIMLSVM | DIVERSE DENSITY | EM-DD |
| *desert* | **.869**±**.014** | .868±.026 | .768±.037 | .751±.047 |
| *mountains* | .791±.024 | **.820**±**.022** | .721±.030 | .717±.036 |
| *sea* | .729±.026 | **.730**±**.030** | .587±.038 | .639±.063 |
| *sunset* | .864±.033 | **.883**±**.023** | .841±.036 | .815±.063 |
| *trees* | **.801**±**.015** | .798±.017 | .781±.028 | .632±.060 |
| *overall* | .811±.022 | **.820**±**.024** | .739±.034 | .711±.054 |

## 5 Conclusion

In this paper, we formalize *multi-instance multi-label learning* where an example is associated with multiple instances and multiple labels simultaneously. Although there were some works investigating the ambiguity of alternative input descriptions or alternative output descriptions associated with an object, this is the first work studying both these ambiguities simultaneously. We show that an MIML problem can be solved by identifying its equivalence in the traditional supervised learning framework, using multi-instance learning or multi-label learning as the bridge. The proposed algorithms, MIMLBOOST and MIMLSVM, have achieved good performance in the application to scene classification. An interesting future issue is to develop MIML versions of other popular machine learning algorithms. Moreover, it remains an open problem that whether MIML can be tackled directly, possibly by exploiting the connections between the instances and the labels. It is also interesting to discover the relationship between the instances and labels. By unravelling the mixed connections, maybe we can get deeper understanding of ambiguity.

### Acknowledgments

This work was supported by the National Science Foundation of China (60325207, 60473046).

## Footnotes

[1] According to notions used in multi-instance learning, $(X_i, y_i)$ is a labeled *bag* while $X_i$ an unlabeled bag.

[2] Although most works on multi-label learning assume that an instance can be associated with multiple valid labels, there are also works assuming that only one of the labels associated with an instance is correct [6]. We adopt the former assumption in this paper.

[3]This constraint assumes that all instances contribute equally and independently to a bag's label, which is different from the standard multi-instance assumption that there is one 'key' instance in a bag that triggers whether the bag's class label will be positive or negative. Nevertheless, it has been shown that this assumption is reasonable and effective [9]. Note that the standard multi-instance assumption does not always hold, e.g. the label *Africa* of an image is usually triggered by several patches jointly instead of by only one patch.

[4]In preliminary experiments, several percentage values have been tested ranging from 20% to 100% with an interval of 20%. The results show that these values do not significantly affect the performance of MIMLSVM.

## References

[1] M. R. Boutell, J. Luo, X. Shen, and C. M. Brown. Learning multi-label scene classification. *Pattern Recognition*, 37(9):1757–1771, 2004.

[2] C.-C. Chang and C.-J. Lin. LIBSVM: A library for support vector machines. Technical report, Department of Computer Science and Information Engineering, National Taiwan University, Taipei, 2001.

[3] Y. Chen and J. Z. Wang. Image categorization by learning and reasoning with regions. *Journal of Machine Learning Research*, 5:913–939, 2004.

[4] T. G. Dietterich, R. H. Lathrop, and T. Lozano-Pérez. Solving the multiple-instance problem with axis-parallel rectangles. *Artificial Intelligence*, 89(1-2):31–71, 1997.

[5] G. A. Edgar. *Measure, Topology, and Fractal Geometry*. Springer, Berlin, 1990.

[6] R. Jin and Z. Ghahramani. Learning with multiple labels. In S. Becker, S. Thrun, and K. Obermayer, editors, *Advances in Neural Information Processing Systems 15*, pages 897–904. MIT Press, Cambridge, MA, 2003.

[7] O. Maron and A. L. Ratan. Multiple-instance learning for natural scene classification. In *Proceedings of the 15th International Conference on Machine Learning*, pages 341–349, Madison, MI, 1998.

[8] R. E. Schapire and Y. Singer. BoosTexter: A boosting-based system for text categorization. *Machine Learning*, 39(2-3):135–168, 2000.

[9] X. Xu and E. Frank. Logistic regression and boosting for labeled bags of instances. In H. Dai, R. Srikant, and C. Zhang, editors, *Lecture Notes in Artificial Intelligence 3056*, pages 272–281. Springer, Berlin, 2004.

[10] Q. Zhang and S. A. Goldman. EM-DD: An improved multi-instance learning technique. In T. G. Dietterich, S. Becker, and Z. Ghahramani, editors, *Advances in Neural Information Processing Systems 14*, pages 1073–1080. MIT Press, Cambridge, MA, 2002.

[11] Z.-H. Zhou and M.-L. Zhang. Solving multi-instance problems with classifier ensemble based on constructive clustering. *Knowledge and Information Systems*, in press.
